# Geometric entropy minimization (GEM) for anomaly detection and localization

**Alfred O Hero, III**
University of Michigan
Ann Arbor, MI 48109-2122
`hero@umich.edu`

## Abstract

We introduce a novel adaptive non-parametric anomaly detection approach, called GEM, that is based on the minimal covering properties of K-point entropic graphs when constructed on N training samples from a nominal probability distribution. Such graphs have the property that as $N \to \infty$ their span recovers the entropy minimizing set that supports at least $\rho = K/N(100)\%$ of the mass of the Lebesgue part of the distribution. When a test sample falls outside of the entropy minimizing set an anomaly can be declared at a statistical level of significance $\alpha = 1 - \rho$. A method for implementing this non-parametric anomaly detector is proposed that approximates this minimum entropy set by the influence region of a K-point entropic graph built on the training data. By implementing an incremental leave-one-out k-nearest neighbor graph on resampled subsets of the training data GEM can efficiently detect outliers at a given level of significance and compute their empirical p-values. We illustrate GEM for several simulated and real data sets in high dimensional feature spaces.

## 1  Introduction

Anomaly detection and localization are important but notoriously difficult problems. In such problems it is crucial to identify a nominal or baseline feature distribution with respect to which statistically significant deviations can be reliably detected. However, in most applications there is seldom enough information to specify the nominal density accurately, especially in high dimensional feature spaces for which the baseline shifts over time. In such cases standard methods that involve estimation of the multivariate feature density from a fixed training sample are inapplicable (high dimension) or unreliable (shifting baseline). In this paper we propose an adaptive non-parametric method that is based on a class of entropic graphs [1] called $K$-point minimal spanning trees [2] and overcomes the limitations of high dimensional feature spaces and baseline shift. This method detects outliers by comparing them to the most concentrated subset of points in the training sample. It follows from [2] that this most concentrated set converges to the minimum entropy set of probability $\rho$ as $N \to \infty$ and $K/N \to \rho$. Thus we call this approach to anomaly detection the geometric entropy minimization (GEM) method.

Several approaches to anomaly detection have been previously proposed. Parametric approaches such as the generalized likelihood ratio test lead to simple and classical algorithms such as the Student t-test for testing deviation of a Gaussian test sample from a nominal mean value and the Fisher F-test for testing deviation of a Gaussian test sample from a nominal variance. These methods fall under the statistical nomenclature of the classical slippage problem [3] and have been applied to detecting abrupt changes in dynamical systems, image segmentation, and general fault detection applications [4]. The main drawback of these algorithms is that they rely on a family of parameterically defined nominal (no-fault) distributions.

An alternative to parametric methods of anomaly detection are the class of novelty detection algorithms and include the GEM approach described herein. Scholkopf and Smola introduced a kernel-based novelty detection scheme that relies on unsupervised support vector machines (SVM) [5]. The single class minimax probability machine of Lanckriet *etal* [6] derives minimax linear decision regions that are robust to unknown anomalous densities. More closely related to our GEM approach is that of Scott and Nowak [7] who derive multiscale approximations of minimum-volume-sets to estimate a particular level set of the unknown nominal multivariate density from training samples. For a simple comparative study of several of these methods in the context of detecting network intrusions the reader is referred to [8].

The GEM method introduced here has several features that are summarized below. (1) Unlike the MPM method of Lanckriet *etal* [6] the GEM anomaly detector is not restricted to linear or even convex decision regions. This translates to higher power for specified false alarm level. (2) GEMs computational complexity scales linearly in dimension and can be applied to level set estimation in feature spaces of unprecedented (high) dimensionality. (3) GEM has no complicated tuning parameters or function approximation classes that must be chosen by the user. (4) Like the method of Scott and Nowak [7] GEM is completely non-parametric, learning the structure of the nominal distribution without assumptions of linearity, smoothness or continuity of the level set boundaries. (5) Like Scott and Nowak's method, GEM is provably optimal - indeed uniformly most powerful of specified level - for the case that the anomaly density is a mixture of the nominal and a uniform density. (6) GEM easily adapts to local structure, e.g. changes in local dimensionality of the support of the nominal density.

We introduce an incremental Leave-one-out (L1O) kNNG as a particularly versatile and fast anomaly detector in the GEM class. Despite the similarity in nomenclature, the L1O kNNG is different from k nearest neighbor (kNN) anomaly detection of [9]. The kNN anomaly detector is based on thresholding the distance from the test point to the k-th nearest neighbor. The L1O kNNG detector computes the change in the topology of the entire kNN graph due to the addition of a test sample and does not use a decision threshold. Furthermore, the parent GEM anomaly detection methodology has proven theoretical properties, e.g. the (restricted) optimality property for uniform mixtures and general consistency properties.

We introduce the statistical framework for anomaly detection in the next section. We then describe the GEM approach in Section . Several simulations are presented n Section 4.

## 2   Statistical framework

The setup is the following. Assume that a training sample $\mathcal{X}_n = \{X_1, \ldots, X_n\}$ of $d$-dimensional vectors $X_i$ is available. Given a new sample $X$ the objective is to declare $X$ to be a "nominal" sample consistent with $\mathcal{X}_n$ or an "anomalous" sample that is significantly different from $\mathcal{X}_n$. This declaration is to be constrained to give as few false positives as possible. To formulate this problem we adopt the standard statistical framework for testing composite hypotheses. Assume that $\mathcal{X}_n$ is an independent identically distributed (i.i.d.) sample from a multivariate density $f_0(x)$ supported on the unit $d$-dimensional cube $[0, 1]^d$. Let $X$ have density $f(x)$. Anomaly detection can be formulated as testing the hypotheses $H_0 : f = f_o$ versus $H_0 : f \neq f_o$ at a prescribed level $\alpha$ of significance $P(\text{declare } H_1 | H_0) \leq \alpha$.

The minimum-volume-set of level $\alpha$ is defined as a set $\Omega_\alpha$ in $\mathbf{R}^d$ which minimizes the volume $|\Omega_\alpha| = \int_{\Omega_\alpha} dx$ subject to the constraint $\int_{\Omega_\alpha} f_0(x)dx \geq 1 - \alpha$. The minimum-entropy-set of level $\alpha$ is defined as a set $\Lambda_\alpha$ in $\mathbf{R}^d$ which minimizes the Rényi entropy $H_\nu(\Lambda_\alpha) = \frac{1}{1-\alpha} \ln \int_{\Lambda_\alpha} f^\nu(x)dx$ subject to the constraint $\int_{\Lambda_\alpha} f_0(x)dx \geq 1 - \alpha$. Here $\nu$ is any real valued parameter between $0 < \nu < 1$. When $f$ is a Lebesgue density in $\mathbf{R}^d$ it is easy to show that these three sets are identical almost everywhere.

The test "decide anomaly if $X \notin \Omega_\alpha$" is equivalent to implementing the test function

$$\phi(x) = \left\{ \begin{array}{ll} 1, & x \notin \Omega_\alpha \\ 0, & o.w. \end{array} \right\}.$$

This test has a strong optimality property: when $f_0$ is Lebesgue continuous it is a uniformly most powerful (UMP) level $\alpha$ for testing anomalies that follow a uniform mixture distribution. Specif-

ically, let $X$ have density $f(x) = (1 - \epsilon)f_0(x) + \epsilon U(x)$ where $U(x)$ is the uniform density over $[0, 1]^d$ and $\epsilon \in [0, 1]$. Consider testing the hypotheses

$$H_0 \quad : \quad \epsilon = 0 \tag{1}$$
$$H_1 \quad : \quad \epsilon > 0 \tag{2}$$

**Proposition 1** *Assume that under $H_0$ the random vector $X$ has a Lebesgue continuous density $f_0$ and that $Z = f_0(X)$ is also a continuous random variable. Then the level-set test of level $\alpha$ is uniformly most powerful for testing (2). Furthermore, its power function $\beta = P(X \notin \Omega_\alpha | H_1)$ is given by*

$$\beta = (1 - \epsilon)\alpha + \epsilon(1 - |\Omega_\alpha|).$$

A sufficient condition for the random variable $Z$ above to be continuous is that the density $f_0(x)$ have no flat spots over its support set $\{f_0(x) > 0\}$. The proof of this proposition is omitted.

There are two difficulties with implementing the level set test. First, for known $f_0$ the level set may be very difficult if not impossible to determine in high dimensions $d \gg 2$. Second, when only a training sample from $f_0$ is available and $f_0$ is unknown the level sets have to be learned from the training data. There are many approaches to doing this for minimum volume tests and these are reviewed in [7]. These methods can be divided into two main approaches: density estimation followed by plug in estimation of $\Omega_\alpha$ via variational methods; and (2) direct estimation of the level set using function approximation and non-parametric estimation. Since both approaches involve explicit approximation of high dimensional quantities, e.g. the multivariate density or the boundary of the set $\Omega\alpha$, these methods are difficult to apply in high dimensional problems, i.e. $d > 2$. The GEM method we propose in the next section overcomes these difficulties.

## 3   GEM and entropic graphs

GEM is a method that directly estimates the critical region for detecting anomalies using minimum coverings of subsets of points in a nominal training sample. These coverings are obtained by constructing minimal graphs, e.g. a MST or kNNG, covering a $K$-point subset that is a given proportion of the training sample. Points not covered by these $K$-point minimal graphs are identified as tail events and allow one to adaptively set a pvalue for the detector.

For a set of $n$ points $\mathcal{X}_n$ in $\mathbf{R}^d$ a graph $\mathcal{G}$ over $\mathcal{X}_n$ is a pair $(V, E)$ where $V = \mathcal{X}_n$ is the set of vertices and $E = \{e\}$ is the set of edges of the graph. The total power weighted length, or, more simply, the length, of $\mathcal{G}$ is $L(\mathcal{X}_n) = \sum_{e \in E} |e|^\gamma$ where $\gamma > 0$ is a specified edge exponent parameter.

### 3.1   K-point MST

The MST with power weighting $\gamma$ is defined as the graph that spans $\mathcal{X}_n$ with minimum total length:

$$L_{MST}(\mathcal{X}_n) = \min_{T \in \mathcal{T}} \sum_{e \in T} |e|^\gamma.$$

where $\mathcal{T}$ is the set of all trees spanning $\mathcal{X}_n$.

**Definition 1   K-point MST**: *Let $\mathcal{X}_{n,K}$ denote one of the $\binom{n}{K}$ subsets of $K$ distinct points from $\mathcal{X}_n$. Among all of the MST's spanning these sets, the K-MST is defined as the one having minimal length $\min_{\mathcal{X}_{n,K} \subset \mathcal{X}_n} L_{MST}(\mathcal{X}_{n,k})$.*

The $K$-MST thus specifies the minimal subset of K points in addition to specifying the minimum length. This subset of points, which we call a minimal graph covering of $\mathcal{X}_n$ of size $K$, can be viewed as capturing the densest region of $\mathcal{X}_n$. Furthermore, if $\mathcal{X}_n$ is a i.i.d. sample from a multivariate density $f(x)$ and if $\lim_{K,n \to \infty} K/n = \rho$ and a greedy version of the $K$-MST is implemented, this set converges a.s. to the minimum $\nu$-entropy set containing a proportion of at least $\rho = K/n$ of the mass of the (Lebesgue component of) $f(x)$, where $\nu = (d - \gamma)/d$. This fact was used in [2] to motivate the greedy $K$-MST as an outlier resistant estimator of entropy for finite $n, K$.

Define the $K$-point subset

$$\mathcal{X}_{n,K}^* = \operatorname{argmin}_{\mathcal{X}_{n,K} \subset \mathcal{X}_n} L_{MST}(\mathcal{X}_{n,K})$$

selected by the greedy K-MST. Then we have the following As the minimum entropy set and minimum volume set are identical, this suggests the following minimal-volume-set anomaly detection algorithm, which we call the "K-MST anomaly detector."

## K-MST anomaly detection algorithm

[1]**Process training sample**: Given a level of significance $\alpha$ and a training sample $\mathcal{X}_n = \{X_1, \ldots, X_n\}$, construct the greedy K-MST and retain its vertex set $\mathcal{X}_{n,K}^*$.

[2]**Process test sample**: Given a test sample $X$ run the K-MST on the merged training-test sample $\mathcal{X}_{n+1} = \mathcal{X}_n \cup \{X\}$ and store the minimal set of points $\mathcal{X}_{n+1,K}^*$.

[3]**Make decision**: Using the test function $\phi$ defined below decide $H_1$ if $\phi(X) = 1$ and decide $H_0$ if $\phi(X) = 0$.

$$\phi(x) = \begin{cases} 1, & X \notin \mathcal{X}_{n+1,K}^* \\ 0, & o.w. \end{cases}.$$

When the density $f_0$ generating the training sample is Lebesgue continuous, it follows from [2, Theorem 2] that as $K, n \to \infty$ the K-MST anomaly detector has false alarm probability that converges to $\alpha = 1 - K/n$ and power that converges to that of the minimum-volume-set test of level $\alpha$. When the density $f_0$ is not Lebesgue continuous some optimality properties of the K-MST anomaly detector still hold. Let this nominal density have the decomposition $f_0 = \lambda_0 + \delta_0$, where $\lambda_0$ is Lebesgue continuous and $\delta_0$ is singular. Then, according to [2, Theorem 2], the K-MST anomaly detector will have false alarm probability that converges to $(1 - \psi)\alpha$, where $\psi$ is the mass of the singular component of $f_0$, and it is a uniformly most powerful test for anomalies in the continuous component, i.e. for the test of $H_0 : \lambda = \lambda_0, \ \delta = \delta_0$ against $H_1 : \lambda = (1 - \epsilon)\lambda_0 + \epsilon U(x), \ \delta = \delta_0$.

It is well known that the K-MST construction is of exponential complexity in $n$ [10]. In fact, even for $K = n - 1$, a case one can call the leave-one-out MST, there is no simple fast algorithm for computation. However, the leave-one-out kNNG, described below, admits a fast incremental algorithm.

### 3.2 K-point kNNG

Let $\mathcal{X}_n = \{X_1, \ldots, X_n\}$ be a set of $n$ points. The k nearest neighbors (kNN) $\{X_{i(1)}, \ldots X_{i(k)}\}$ of a point $X_i \in \mathcal{X}_n$ are the $k$ closest points to $X_i$ points in $\mathcal{X}_n - \{X_i\}$. Here the measure of closeness is the Euclidean distance. Let $\{e_{i(1)}, \ldots, e_{i(k)}\}$ be the set of edges between $X_i$ and its $k$ nearest neighbors. The kNN graph (kNNG) over $\mathcal{X}_n$ is defined as the union of all of the kNN edges $\{e_{i(1)}, \ldots, e_{i(k)}\}_{i=1}^n$ and the total power weighted edge length of the kNN graph is

$$L_{kNN}(\mathcal{X}_n) = \sum_{i=1}^{n} \sum_{l=1}^{k} |e_{i(l)}|^{\gamma}.$$

**Definition 2 K-point kNNG**: *Let $\mathcal{X}_{n,K}$ denote one of the $\binom{n}{K}$ subsets of $K$ distinct points from $\mathcal{X}_n$. Among all of the kNNG over each of these sets, the K-kNNG is defined as the one having minimal length $\min_{\mathcal{X}_{n,K} \subset \mathcal{X}_n} L_{kNN}(\mathcal{X}_{n,k})$.*

As the kNNG length is also a quasi additive continuous functional [11], the asymptotic KMST theory of [2] extends to the K-point kNNG. Of course, computation of the K-point kNNG also has exponential complexity. However, the same type of greedy approximation introduced by Ravi [10] for the $K$-MST can be implemented to reduce complexity of the K-point kNNG. This approximation to the K-point kNNG will satisfy the tightly coverable graph property of [2, Defn. 2]. We have the following result that justifies the use of such an approximation as an anomaly detector of level $\alpha = 1 - \rho$, where $\rho = K/n$:

**Proposition 2** *Let $\mathcal{X}_{n,K}^*$ be the set of points in $\mathcal{X}_n$ that results from any approximation to the K-point kNNG that satisfies the property [2, Defn. 2]. Then $\lim_{n\to\infty} P_0(\mathcal{X}_{n,K}^* \subset \Omega_\alpha) \to 1$ and $\lim_{n\to\infty} P_0(\mathcal{X}_{n,K}^* \cap \overline{\Omega}_\alpha) \to 0$, where $K = K(n) = \text{floor}(\rho n)$, $\Omega_\alpha$ is a minimum-volume-set of level $\alpha = 1 - \rho$ and $\overline{\Omega}_\alpha = [0, 1]^d - \Omega_\alpha$.*

**Proof**: We provide a rough sketch using the terminology of [2]. Recall that a set $B^m \subset [0,1]^d$ of resolution $1/m$ is representable by a union of elements of the uniform partition of $[0,1]^d$ into hypercubes of volume $1/m^d$. Lemma 3 of [2] asserts that there exists an $M$ such that for $m > M$ the limits claimed in Proposition 2 hold with $\Omega_\alpha$ replaced by $A_\alpha^m$, a minimum volume set of resolution $1/m$ that contains $\Omega_\alpha$. As $\lim_{m \to \infty} A_\rho^m = \Omega_\alpha$ this establishes the proposition.

Figures 1-2 illustrate the use of the K-point kNNG as an anomaly detection algorithm.

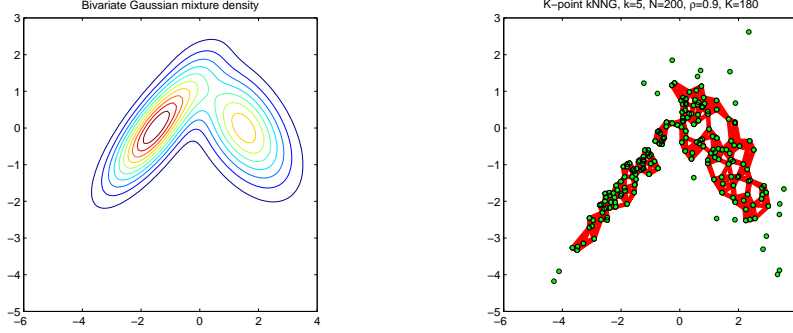

Figure 1: *Left: level sets of the nominal bivariate mixture density used to illustrate the K point kNNG anomaly detection algorithms. Right: K-point kNNG over N=200 random training samples drawn from the nominal bivariate mixture at left. Here k=5 and K=180, corresponding to a significance level of $\alpha = 0.1$.*

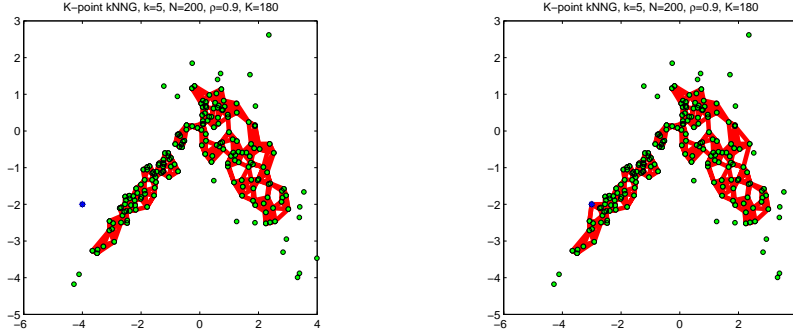

Figure 2: *Left: The test point '\*' is declared anomalous at level $\alpha = 0.1$ as it is not captured by the K-point kNNG (K=180) constructed over the combined test sample and the training samples drawn from the nominal bivariate mixture shown in Fig. 1. Right: A different test point '\*' is declared non-anomalous as it is captured by this K-point kNNG.*

### 3.3 Leave-one-out kNNG (L1O-kNNG)

The theoretical equivalence between the K-point kNNG and the level set anomaly detector motivates a low complexity anomaly detection scheme, which we call the leave-one-out kNNG, discussed in this section and adopted for the experiments below. As before, assume a single test sample $X = X_{n+1}$ and a training sample $\mathcal{X}_n$. Fix $k$ and assume that the kNNG over the set $\mathcal{X}_n$ has been computed. To determine the kNNG over the combined sample $\mathcal{X}_{n+1} = \mathcal{X}_n \cup \{X_{n+1}\}$ one can execute the following algorithm:

**L1O kNNG anomaly detection algorithm**

1. For each $X_i \in \mathcal{X}_{n+1}$, $i = 1, \ldots, n+1$, compute the kNNG total length difference $\Delta_i L_{kNN} = L_{kNN}(\mathcal{X}_{n+1}) - L_{kNN}(\mathcal{X}_{n+1} - \{X_i\})$ by the following steps. For each $i$:

(a) Find the $k$ edges $\mathcal{E}^k_{i\to*}$ of all of the kNN's of $X_i$.

(b) Find the edges $\mathcal{E}^k_{*\to i}$ of other points in $\mathcal{X}_{n+1} - \{X_i\}$ that have $X_i$ as one of their kNNs. For these points find the edges $\mathcal{E}^{k+1}_*$ to their respective $k+1$st NN point.

(c) Compute $\Delta_i L_{kNN} = \sum_{e\in\mathcal{E}^k_{i\to*}} |e|^\gamma + \sum_{e\in\mathcal{E}^k_{*\to i}} |e|^\gamma - \sum_{e\in\mathcal{E}^{k+1}_*} |e|^\gamma$

2. Define the kNNG most "outlying point" as $X_o = \text{argmax}_{i=1,\dots,n+1}\Delta_i L_{kNN}$.

3. Declare the test sample $X_{n+1}$ an anomaly if $X_{n+1} = X_o$.

This algorithm will detect anomalies with a false alarm level of approximately $1/(n+1)$. Thus larger sizes $n$ of the training sample will correspond to more stringent false alarm constraints. Furthermore, the p-value of each test point $X_i$ is easily computed by recursing over the size $n$ of the training sample. In particular, let $n'$ vary from $k$ to $n$ and define $n^*$ as the minimum value of $n'$ for which $X_i$ is declared an anomaly. Then the p-value of $X_i$ is approximately $1/(n^*+1)$.

A useful relative influence coefficient $\eta$ can be defined for each point $X_i$ in the combined sample $\mathcal{X}_{n+1}$

$$\eta(X_i) = \frac{\Delta_i L_{kNN}}{\max_i \Delta_i L_{kNN}}. \tag{3}$$

The coefficient $\eta(X_{n+1}) = 1$ when the test point $X_{n+1}$ is declared an anomaly.

Using matlab's matrix sort algorithm step 1 of this algorithm can be computed an order of magnitude faster than the K-point MST ($N^2 logN$ vs $N^3 logN$). For example, the experiments below have shown that the above algorithm can find and determine the p-value of 10 outliers among 1000 test samples in a few seconds on a Dell 2GHz processor running Matlab 7.1.

## 4 Illustrative examples

Here we focus on the L1O kNNG algorithm due to its computational speed. We show a few representative experiments for simple Gaussian and Gaussian mixture nominal densities $f_0$.

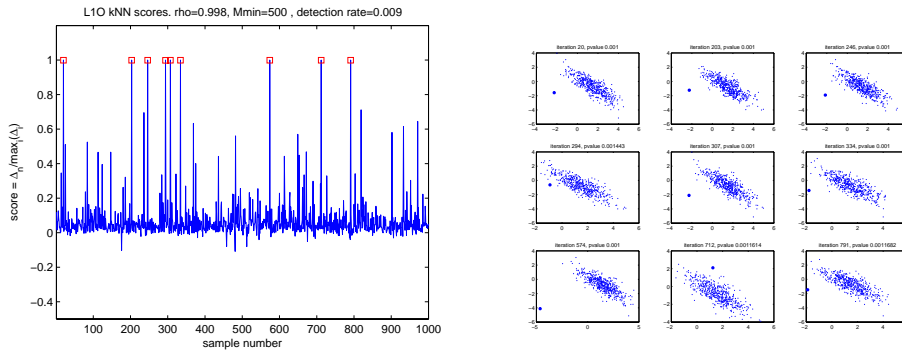

Figure 3: *Left: The plot of the anomaly curve for the L1O kNNG anomaly detector for detecting deviations from a nominal 2D Gaussian density with mean (0,0) and correlation coefficient -0.5. The boxes on peaks of curve correspond to positions of detected anomalies and the height of the boxes are equal to one minus the computed p-value. Anomalies were generated (on the average) every 100 samples and drawn from a 2D Gaussian with correlation coefficient 0.8. The parameter $\rho$ is equal to $1 - \alpha$, where $\alpha$ is the user defined false alarm rate. Right: the resampled nominal distribution ("•") and anomalous points detected ("*") at the iterations indicated at left.*

First we illustrate the L1O kNNG algorithm for detection of non-uniformly distributed anomalies from training samples following a bivariate Gaussian nominal density. Specifically, a 2D Gaussian density with mean (0,0) and correlation coefficient -0.5 was generated to train of the L1O kNNG detector. The test sample consisted of a mixture of this nominal and a zero mean 2D Gaussian with correlation coefficient 0.8 with mixture coefficient $\epsilon = 0.01$. In Fig. 3 the results of simulation with a training sample of 2000 samples and 1000 tests samples are shown. Fig. 3 is a plot of the relative

influence curve (3) over the test samples as compared to the most outlying point in the (resampled) training sample. When the relative influence curve is equal to 1 the corresponding test sample is the most outlying point and is declared an anomaly. The 9 detected anomalies in Fig. 3 have p-values less than 0.001 and therefore one would expect an average of only one false alarm at this level of significance. In the right panel of Fig. 3 the detected anomalies (asterisks) are shown along with the training sample (dots) used to grow the L1O kNNG for that particular iteration - note that to protect against bias the training sample is resampled at each iteration.

Next we compare the performance of the L1O kNNG detector to that of the UMP test for the hypotheses (2). We again trained on a bivariate Gaussian $f_0$ with mean zero, but this time with identical component variances of $\sigma = 0.1$. This distribution has essential support on the unit square. For this simple case the minimum volume set of level $\alpha$ is a disk centered at the origin with radius $\sqrt{2\sigma^2 \ln 1/\alpha}$ and the power of the of the UMP can be computed in closed form: $\beta = (1 - \epsilon)\alpha + \epsilon(1 - 2\pi\sigma^2 \ln 1/\alpha)$. We implemented the GEM anomaly detector with the incremental leave-one-out kNNG using $k = 5$. The training set consisted of 1000 samples from $f_0$ and the test set consisted of 1000 samples from the mixture of a uniform density and $f_0$ with parameter $\epsilon$ ranging from 0 to 0.2. Figure 4 shows the empirical ROC curves obtained using the GEM test vs the theoretical curves (labeled "clairvoyant") for several different values of the mixing parameter. Note the good agreement between theoretical prediction and the GEM implementation of the UMP using the kNNG.

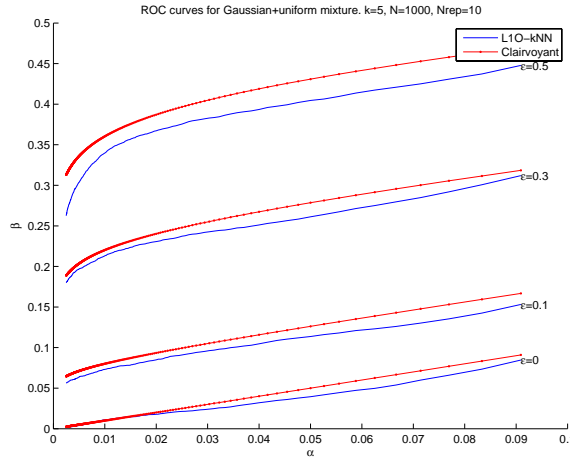

Figure 4: *ROC curves for the leave-one-out kNNG anomaly detector described in Sec. 3.3. The labeled "clairvoyant" curve is the ROC of the UMP anomaly detector. The training sample is a zero mean 2D spherical Gaussian distribution with standard deviation 0.1 and the test sample is a this 2D Gaussian and a 2D uniform-$[0, 1]^2$ mixture density. The plot is for various values of the mixture parameter $\epsilon$.*

## 5   Conclusions

A new and versatile anomaly detection method has been introduced that uses geometric entropy minimization (GEM) to extract minimal set coverings that can be used to detect anomalies from a set of training samples. This method can be implemented through the K-point minimal spanning tree (MST) or the K-point nearest neighbor graph (kNNG). The L1O kNNG is significantly less computationally demanding than the K-point MST. We illustrated the L1O kNNG method on simulated data containing anomalies and showed that it comes close to achieving the optimal performance of the UMP detector for testing the nominal against a uniform mixture with unknown mixing parameter. As the L1O kNNG computes p-values on detected anomalies it can be easily extended to account for false discovery rate constraints. By using a sliding window, the methodology derived in this paper is easily extendible to on-line applications and has been applied to non-parametric intruder detection using our Crossbow sensor network testbed (reported elsewhere).

**Acknowledgments**

This work was partially supported by NSF under Collaborative ITR grant CCR-0325571.

# References

[1] A. Hero, B. Ma, O. Michel, and J. Gorman, "Applications of entropic spanning graphs," *IEEE Signal Processing Magazine*, vol. 19, pp. 85–95, Sept. 2002. `www.eecs.umich.edu/~hero/imag_proc.html`.

[2] A. Hero and O. Michel, "Asymptotic theory of greedy approximations to minimal k-point random graphs," *IEEE Trans. on Inform. Theory*, vol. IT-45, no. 6, pp. 1921–1939, Sept. 1999.

[3] T. S. Ferguson, *Mathematical Statistics - A Decision Theoretic Approach*. Academic Press, Orlando FL, 1967.

[4] I. V. Nikiforov and M. Basseville, *Detection of abrupt changes: theory and applications*. Prentice-Hall, Englewood-Cliffs, NJ, 1993.

[5] B. Scholkopf, R. Williamson, A. Smola, J. Shawe-Taylor, and J. Platt, "Support vector method for novelty detection," in *Advances in Neural Information Processing Systems (NIPS)*, vol. 13, 2000.

[6] G. R. G. Lanckriet, L. El Ghaoui, and M. I. Jordan, "Robust novelty detection with single-class mpm," in *Advances in Neural Information Processing Systems (NIPS)*, vol. 15, 2002.

[7] C. Scott and R. Nowak, "Learning minimum volume sets," *Journal of Machine Learning Research*, vol. 7, pp. 665–704, April 2006.

[8] A. Lazarevic, A. Ozgur, L. Ertoz, J. Srivastava, and V. Kumar, "A comparative study of anomaly detection schemes in network intrusion detection," in *SIAM Conference on data mining*, 2003.

[9] S. Ramaswamy, R. Rastogi, and K. Shim, "Efficient algorithms for mining outliers from large data sets," in *Proceedings of the ACM SIGMOD Conference*, 2000.

[10] R. Ravi, M. Marathe, D. Rosenkrantz, and S. Ravi, "Spanning trees short or small," in *Proc. 5th Annual ACM-SIAM Symposium on Discrete Algorithms*, (Arlington, VA), pp. 546–555, 1994.

[11] J. E. Yukich, *Probability theory of classical Euclidean optimization*, vol. 1675 of *Lecture Notes in Mathematics*. Springer-Verlag, Berlin, 1998.
